# Generalized Nonnegative Matrix Approximations with Bregman Divergences

**Inderjit S. Dhillon**   **Suvrit Sra**
Dept. of Computer Sciences
The Univ. of Texas at Austin
Austin, TX 78712.
{inderjit,suvrit}@cs.utexas.edu

## Abstract

Nonnegative matrix approximation (NNMA) is a recent technique for dimensionality reduction and data analysis that yields a parts based, sparse nonnegative representation for nonnegative input data. NNMA has found a wide variety of applications, including text analysis, document clustering, face/image recognition, language modeling, speech processing and many others. Despite these numerous applications, the algorithmic development for computing the NNMA factors has been relatively deficient. This paper makes algorithmic progress by modeling and *solving* (using multiplicative updates) new generalized NNMA problems that minimize Bregman divergences between the input matrix and its low-rank approximation. The multiplicative update formulae in the pioneering work by Lee and Seung [11] arise as a special case of our algorithms. In addition, the paper shows how to use penalty functions for incorporating constraints other than nonnegativity into the problem. Further, some interesting extensions to the use of "link" functions for modeling nonlinear relationships are also discussed.

## 1   Introduction

Nonnegative matrix approximation (NNMA) is a method for dimensionality reduction and data analysis that has gained favor over the past few years. NNMA has previously been called *positive matrix factorization* [13] and *nonnegative matrix factorization*[1] [12]. Assume that $\boldsymbol{a}_1, \ldots, \boldsymbol{a}_N$ are $N$ nonnegative input ($M$-dimensional) vectors. We organize these vectors as the columns of a nonnegative data matrix

$$\boldsymbol{A} \triangleq \begin{bmatrix} \boldsymbol{a}_1 & \boldsymbol{a}_2 & \ldots & \boldsymbol{a}_N \end{bmatrix}.$$

NNMA seeks a small set of $K$ nonnegative representative vectors $\boldsymbol{b}_1, \ldots, \boldsymbol{b}_K$ that can be nonnegatively (or conically) combined to approximate the input vectors $\boldsymbol{a}_i$. That is,

$$\boldsymbol{a}_n \approx \sum_{k=1}^{K} c_{kn} \boldsymbol{b}_k, \quad 1 \le n \le N,$$

where the combining coefficients $c_{kn}$ are restricted to be nonnegative. If $c_{kn}$ and $\boldsymbol{b}_k$ are unrestricted, and we minimize $\sum_n \|\boldsymbol{a}_n - \boldsymbol{B}\boldsymbol{c}_n\|^2$, the Truncated Singular Value Decomposition (TSVD) of $\boldsymbol{A}$ yields the optimal $\boldsymbol{b}_k$ and $c_{kn}$ values. If the $\boldsymbol{b}_k$ are unrestricted, but the coefficient vectors $\boldsymbol{c}_n$ are restricted to be indicator vectors, then we obtain the problem of hard-clustering (See [16, Chapter 8] for related discussion regarding different constraints on $\boldsymbol{c}_n$ and $\boldsymbol{b}_k$).

In this paper we consider problems where all involved matrices are nonnegative. For many practical problems nonnegativity is a natural requirement. For example, color intensities, chemical concentrations, frequency counts etc., are all nonnegative entities, and approximating their measurements by nonnegative representations leads to greater interpretability. NNMA has found a significant number of applications, not only due to increased interpretability, but also because admitting only nonnegative combinations of the $\boldsymbol{b}_k$ leads to sparse representations.

This paper contributes to the algorithmic advancement of NNMA by generalizing the problem significantly, and by deriving efficient algorithms based on multiplicative updates for the generalized problems. The scope of this paper is primarily on generic methods for NNMA, rather than on specific applications. The multiplicative update formulae in the pioneering work by Lee and Seung [11] arise as a special case of our algorithms, which seek to minimize Bregman divergences between the nonnegative input $\boldsymbol{A}$ and its approximation. In addition, we discuss the use penalty functions for incorporating constraints other than nonnegativity into the problem. Further, we illustrate an interesting extension of our algorithms for handling non-linear relationships through the use of "link" functions.

## 2   Problems

Given a nonnegative matrix $\boldsymbol{A}$ as input, the classical NNMA problem is to approximate it by a lower rank nonnegative matrix of the form $\boldsymbol{B}\boldsymbol{C}$, where $\boldsymbol{B} = [\boldsymbol{b}_1, ..., \boldsymbol{b}_K]$ and $\boldsymbol{C} = [\boldsymbol{c}_1, ..., \boldsymbol{c}_N]$ are themselves nonnegative. That is, we seek the approximation,

$$\boldsymbol{A} \approx \boldsymbol{B}\boldsymbol{C}, \qquad \text{where } \boldsymbol{B}, \boldsymbol{C} \geq 0. \tag{2.1}$$

We judge the goodness of the approximation in (2.1) by using a general class of distortion measures called *Bregman divergences*. For any strictly convex function $\varphi : S \subseteq \mathbb{R} \to \mathbb{R}$ that has a continuous first derivative, the corresponding **Bregman divergence** $D_\varphi : S \times \text{int}(S) \to \mathbb{R}_+$ is defined as $D_\varphi(x, y) \triangleq \varphi(x) - \varphi(y) - \nabla\varphi(y)(x - y)$, where $\text{int}(S)$ is the interior of set $S$ [1, 2]. Bregman divergences are nonnegative, convex in the first argument and zero if and only if $x = y$. These divergences play an important role in convex optimization [2]. For the sequel we consider only separable Bregman divergences, i.e., $D_\varphi(\boldsymbol{X}, \boldsymbol{Y}) = \sum_{ij} D_\varphi(x_{ij}, y_{ij})$. We further require $x_{ij}, y_{ij} \in \text{dom}\varphi \cap \mathbb{R}_+$.

Formally, the resulting generalized nonnegative matrix approximation problems are:

$$\min_{\boldsymbol{B}, \boldsymbol{C} \geq 0} \quad D_\varphi(\boldsymbol{B}\boldsymbol{C}, \boldsymbol{A}) + \alpha(\boldsymbol{B}) + \beta(\boldsymbol{C}), \tag{2.2}$$

$$\min_{\boldsymbol{B}, \boldsymbol{C} \geq 0} \quad D_\varphi(\boldsymbol{A}, \boldsymbol{B}\boldsymbol{C}) + \alpha(\boldsymbol{B}) + \beta(\boldsymbol{C}). \tag{2.3}$$

The functions $\alpha$ and $\beta$ serve as *penalty* functions, and they allow us to enforce regularization (or other constraints) on $\boldsymbol{B}$ and $\boldsymbol{C}$. We consider both (2.2) and (2.3) since Bregman divergences are generally asymmetric. Table 1 gives a small sample of NNMA problems to illustrate the breadth of our formulation.

## 3   Algorithms

In this section we present algorithms that seek to optimize (2.2) and (2.3). Our algorithms are iterative in nature, and are directly inspired by the efficient algorithms of Lee and Seung [11]. Appealing properties include ease of implementation and computational efficiency.

| Divergence $D_\varphi$ | $\varphi$ | $\alpha$ | $\beta$ | Remarks |
|---|---|---|---|---|
| $\|\boldsymbol{A} - \boldsymbol{BC}\|_\mathrm{F}^2$ | $\frac{1}{2}x^2$ | $\mathbf{0}$ | $\mathbf{0}$ | Lee and Seung [11, 12] |
| $\|\boldsymbol{A} - \boldsymbol{BC}\|_\mathrm{F}^2$ | $\frac{1}{2}x^2$ | $\mathbf{0}$ | $\lambda\mathbf{1}^T\boldsymbol{C}\mathbf{1}$ | Hoyer [10] |
| $\|\boldsymbol{W} \odot (\boldsymbol{A} - \boldsymbol{BC})\|_\mathrm{F}^2$ | $\frac{1}{2}x^2$ | $\mathbf{0}$ | $\mathbf{0}$ | Paatero and Tapper [13] |
| $\mathrm{KL}(\boldsymbol{A}, \boldsymbol{BC})$ | $x \log x$ | $\mathbf{0}$ | $\mathbf{0}$ | Lee and Seung [11] |
| $\mathrm{KL}(\boldsymbol{A}, \boldsymbol{WBC})$ | $x \log x$ | $\mathbf{0}$ | $\mathbf{0}$ | Guillamet et al. [9] |
| $\mathrm{KL}(\boldsymbol{A}, \boldsymbol{BC})$ | $x \log x$ | $c\mathbf{1}\boldsymbol{B}^T\boldsymbol{B}\mathbf{1}$ | $-c'\|\boldsymbol{C}\|_\mathrm{F}^2$ | Feng et al. [8] |
| $D_\varphi(\boldsymbol{A}, \boldsymbol{W}_1\boldsymbol{BC}\boldsymbol{W}_2)$ | $\varphi(x)$ | $\alpha(\boldsymbol{B})$ | $\beta(\boldsymbol{C})$ | Weighted NNMA (new) |

Table 1: Some example NNMA problems that may be obtained from (2.3). The corresponding asymmetric problem (2.2) has not been previously treated in the literature. $\mathrm{KL}(x, y)$ denotes the generalized KL-Divergence $= \sum_i x_i \log \frac{x_i}{y_i} - x_i + y_i$ (also called I-divergence).

Note that the problems (2.2) and (2.3) are not jointly convex in $\boldsymbol{B}$ and $\boldsymbol{C}$, so it is not easy to obtain globally optimal solutions in polynomial time. Our iterative procedures start by initializing $\boldsymbol{B}$ and $\boldsymbol{C}$ randomly or otherwise. Then, $\boldsymbol{B}$ and $\boldsymbol{C}$ are alternately updated until there is no further appreciable change in the objective function value.

### 3.1 Algorithms for (2.2)

We utilize the concept of auxiliary functions [11] for our derivations. It is sufficient to illustrate our methods using a single column of $\boldsymbol{C}$ (or row of $\boldsymbol{B}$), since our divergences are separable.

**Definition 3.1 (Auxiliary function).** A function $G(\boldsymbol{c}, \boldsymbol{c}')$ is called an auxiliary function for $F(\boldsymbol{c})$ if:

1. $G(\boldsymbol{c}, \boldsymbol{c}) = F(\boldsymbol{c})$, and

2. $G(\boldsymbol{c}, \boldsymbol{c}') \geq F(\boldsymbol{c})$ for all $\boldsymbol{c}'$.

Auxiliary functions turn out to be useful due to the following lemma.

**Lemma 3.2 (Iterative minimization).** *If $G(\boldsymbol{c}, \boldsymbol{c}')$ is an auxiliary function for $F(\boldsymbol{c})$, then $F$ is non-increasing under the update*

$$\boldsymbol{c}^{t+1} = \mathrm{argmin}_{\boldsymbol{c}}\, G(\boldsymbol{c}, \boldsymbol{c}^t).$$

*Proof.* $F(\boldsymbol{c}^{t+1}) \leq G(\boldsymbol{c}^{t+1}, \boldsymbol{c}^t) \leq G(\boldsymbol{c}^t, \boldsymbol{c}^t) = F(\boldsymbol{c}^t)$. $\square$

As can be observed, the sequence formed by the iterative application of Lemma 3.2 leads to a monotonic decrease in the objective function value $F(\boldsymbol{c})$. For an algorithm that iteratively updates $\boldsymbol{c}$ in its quest to minimize $F(\boldsymbol{c})$, the method for proving convergence boils down to the construction of an appropriate auxiliary function. Auxiliary functions have been used in many places before, see for example [5, 11].

We now construct simple auxiliary functions for (2.2) that yield multiplicative updates. To avoid clutter we drop the functions $\alpha$ and $\beta$ from (2.2), noting that our methods can easily be extended to incorporate these functions.

Suppose $\boldsymbol{B}$ is fixed and we wish to compute an updated column of $\boldsymbol{C}$. We wish to minimize

$$F(\boldsymbol{c}) = D_\varphi(\boldsymbol{Bc}, \boldsymbol{a}), \qquad (3.1)$$

where $\boldsymbol{a}$ is the column of $\boldsymbol{A}$ corresponding to the column $\boldsymbol{c}$ of $\boldsymbol{C}$. The lemma below shows how to construct an auxiliary function for (3.1). For convenience of notation we use $\psi$ to denote $\nabla\varphi$ for the rest of this section.

**Lemma 3.3 (Auxiliary function).** *The function*

$$G(\boldsymbol{c}, \boldsymbol{c}') = \sum_{ij} \lambda_{ij} \varphi\left(\frac{b_{ij}c_j}{\lambda_{ij}}\right) - \left(\sum_i \varphi(a_i) + \psi(a_i)\big((\boldsymbol{B}\boldsymbol{c})_i - a_i\big)\right), \qquad (3.2)$$

*with $\lambda_{ij} = (b_{ij}c_j')/(\sum_l b_{il}c_l')$, is an auxiliary function for* (3.1). *Note that by definition $\sum_j \lambda_{ij} = 1$, and as both $b_{ij}$ and $c_j'$ are nonnegative, $\lambda_{ij} \geq 0$.*

*Proof.* It is easy to verify that $G(\boldsymbol{c}, \boldsymbol{c}) = F(\boldsymbol{c})$, since $\sum_j \lambda_{ij} = 1$. Using the convexity of $\varphi$, we conclude that if $\sum_j \lambda_{ij} = 1$ and $\lambda_{ij} \geq 0$, then

$$\begin{aligned}
F(\boldsymbol{c}) &= \sum_i \varphi\left(\sum_j b_{ij}c_j\right) - \varphi(a_i) - \psi(a_i)\big((\boldsymbol{B}\boldsymbol{c})_i - a_i\big) \\
&\leq \sum_{ij} \lambda_{ij}\varphi\left(\frac{b_{ij}c_j}{\lambda_{ij}}\right) - \left(\sum_i \varphi(a_i) + \psi(a_i)\big((\boldsymbol{B}\boldsymbol{c})_i - a_i\big)\right) \\
&= G(\boldsymbol{c}, \boldsymbol{c}').
\end{aligned}$$

$\square$

To obtain the update, we minimize $G(\boldsymbol{c}, \boldsymbol{c}')$ w.r.t. $\boldsymbol{c}$. Let $\psi(\boldsymbol{x})$ denote the vector $[\psi(x_1), \ldots, \psi(x_n)]^T$. We compute the partial derivative

$$\begin{aligned}
\frac{\partial G}{\partial c_p} &= \sum_i \lambda_{ip}\psi\left(\frac{b_{ip}c_p}{\lambda_{ip}}\right)\frac{b_{ip}}{\lambda_{ip}} - \sum_i b_{ip}\psi(a_i) \\
&= \sum_i b_{ip}\psi\left(\frac{c_p}{c_p'}(\boldsymbol{B}\boldsymbol{c}')_i\right) - (\boldsymbol{B}^T\psi(\boldsymbol{a}))_p.
\end{aligned} \qquad (3.3)$$

We need to solve (3.3) for $c_p$ by setting $\partial G/\partial c_p = 0$. Solving this equation analytically is not always possible. However, for a broad class of functions, we can obtain an analytic solution. For example, if $\psi$ is multiplicative (i.e., $\psi(xy) = \psi(x)\psi(y)$) we obtain the following iterative update relations for $\boldsymbol{b}$ and $\boldsymbol{c}$ (see [7])

$$b_p \leftarrow \quad b_p \cdot \psi^{-1}\left(\frac{[\psi(\boldsymbol{a}^T)\boldsymbol{C}^T]_p}{[\psi(\boldsymbol{b}^T\boldsymbol{C})\boldsymbol{C}^T]_p}\right), \qquad (3.4)$$

$$c_p \leftarrow \quad c_p \cdot \psi^{-1}\left(\frac{[\boldsymbol{B}^T\psi(\boldsymbol{a})]_p}{[\boldsymbol{B}^T\psi(\boldsymbol{B}\boldsymbol{c})]_p}\right). \qquad (3.5)$$

It turns out that when $\varphi$ is a convex function of Legendre type, then $\psi^{-1}$ can be obtained by the derivative of the conjugate function $\varphi^*$ of $\varphi$, i.e., $\psi^{-1} = \nabla\varphi^*$ [14].

**Note.** (3.4) & (3.5) coincide with updates derived by Lee and Seung [11], if $\varphi(x) = \frac{1}{2}x^2$.

### 3.1.1 Examples of New NNMA Problems

We illustrate the power of our generic auxiliary functions given above for deriving algorithms with multiplicative updates for some specific interesting problems.

First we consider the problem that seeks to minimize the divergence,

$$\text{KL}(\boldsymbol{B}\boldsymbol{c}, \boldsymbol{a}) \;=\; \sum_i (\boldsymbol{B}\boldsymbol{c})_i \log \frac{(\boldsymbol{B}\boldsymbol{c})_i}{a_i} - (\boldsymbol{B}\boldsymbol{c})_i + a_i, \qquad \boldsymbol{B}, \boldsymbol{c} \geq 0. \qquad (3.6)$$

Let $\varphi(x) = x \log x - x$. Then, $\psi(x) = \log x$, and as $\psi(xy) = \psi(x) + \psi(y)$, upon substituting in (3.3), and setting the resultant to zero we obtain

$$\frac{\partial G}{\partial c_p} = \sum_i b_{ip} \log(c_p(\boldsymbol{Bc'})_i / c'_p) - \sum_i b_{ip} \log a_i = 0,$$

$$\implies (\boldsymbol{B}^T \mathbf{1})_p \log \frac{c_p}{c'_p} = [\boldsymbol{B}^T \log \boldsymbol{a} - \boldsymbol{B}^T \log(\boldsymbol{Bc'})]_p$$

$$\implies c_p = c'_p \cdot \exp\left( \frac{[\boldsymbol{B}^T \log(\boldsymbol{a}/(\boldsymbol{Bc'}))]_p}{[\boldsymbol{B}^T \mathbf{1}]_p} \right).$$

The update for $\boldsymbol{b}$ can be derived similarly.

**Constrained NNMA.** Next we consider NNMA problems that have additional constraints. We illustrate our ideas on a problem with linear constraints.

$$\min_{\boldsymbol{x}} \quad D_\varphi(\boldsymbol{Bc}, \, \boldsymbol{a})$$
$$\text{s.t.} \quad \boldsymbol{Pc} \le \mathbf{0}, \quad \boldsymbol{c} \ge \mathbf{0}. \tag{3.7}$$

We can solve (3.7) problem using our method by making use of an appropriate (differentiable) penalty function that enforces $\boldsymbol{Pc} \le \mathbf{0}$. We consider,

$$F(\boldsymbol{c}) = D_\varphi(\boldsymbol{Bc}, \, \boldsymbol{a}) + \rho \| \max(0, \boldsymbol{Pc}) \|^2, \tag{3.8}$$

where $\rho > 0$ is some penalty constant. Assuming multiplicative $\psi$ and following the auxiliary function technique described above, we obtain the following updates for $\boldsymbol{c}$,

$$c_k \leftarrow c_k \cdot \psi^{-1}\left( \frac{[\boldsymbol{B}^T \psi(\boldsymbol{a})]_k - \rho [\boldsymbol{P}^T (\boldsymbol{Pc})^+]_k}{[\boldsymbol{B}^T \psi(\boldsymbol{Bc})]_k} \right),$$

where $(\boldsymbol{Pc})^+ = \max(\mathbf{0}, \boldsymbol{Pc})$. Note that care must be taken to ensure that the addition of this penalty term does not violate the nonnegativity of $\boldsymbol{c}$, and to ensure that the argument of $\psi^{-1}$ lies in its domain.

**Remarks.** Incorporating additional constraints into (3.6) is however easier, since the exponential updates ensure nonnegativity. Given $\boldsymbol{a} = \mathbf{1}$, with appropriate penalty functions, our solution to (3.6) can be utilized for maximizing entropy of $\boldsymbol{Bc}$ subject to linear or non-linear constraints on $\boldsymbol{c}$.

**Nonlinear models with "link" functions.** If $\boldsymbol{A} \approx h(\boldsymbol{BC})$, where $h$ is a "link" function that models a nonlinear relationship between $\boldsymbol{A}$ and the approximant $\boldsymbol{BC}$, we may wish to minimize $D_\varphi(h(\boldsymbol{BC}), \, \boldsymbol{A})$. We can easily extend our methods to handle this case for appropriate $h$. Recall that the auxiliary function that we used, depended upon the convexity of $\varphi$. Thus, if $(\varphi \circ h)$ is a convex function, whose derivative $\nabla(\varphi \circ h)$ is "factorizable," then we can easily derive algorithms for this problem with link functions. We exclude explicit examples for lack of space and refer the reader to [7] for further details.

## 3.2 Algorithms using KKT conditions

We now derive efficient multiplicative update relations for (2.3), and these updates turn out to be simpler than those for (2.2). To avoid clutter, we describe our methods with $\alpha \equiv 0$, and $\beta \equiv 0$, noting that if $\alpha$ and $\beta$ are differentiable, then it is easy to incorporate them in our derivations. For convenience we use $\zeta(x)$ to denote $\nabla^2(x)$ for the rest of this section.

Using matrix algebra, one can show that the gradients of $D_\varphi(\boldsymbol{A}, \, \boldsymbol{BC})$ w.r.t. $\boldsymbol{B}$ and $\boldsymbol{C}$ are,

$$\nabla_{\boldsymbol{B}} D_\varphi(\boldsymbol{A}, \, \boldsymbol{BC}) = \big(\zeta(\boldsymbol{BC}) \odot (\boldsymbol{BC} - \boldsymbol{A})\big) \boldsymbol{C}^T$$
$$\nabla_{\boldsymbol{C}} D_\varphi(\boldsymbol{A}, \, \boldsymbol{BC}) = \boldsymbol{B}^T \big(\zeta(\boldsymbol{BC}) \odot (\boldsymbol{BC} - \boldsymbol{A})\big),$$

where $\odot$ denotes the elementwise or Hadamard product, and $\zeta$ is applied elementwise to $\boldsymbol{BC}$. According to the KKT conditions, there exist Lagrange multiplier matrices $\boldsymbol{\Lambda} \geq 0$ and $\boldsymbol{\Omega} \geq 0$ such that

$$\nabla_{\boldsymbol{B}} D_\varphi(\boldsymbol{A},\ \boldsymbol{BC}) = \boldsymbol{\Lambda}, \qquad \nabla_{\boldsymbol{C}} D_\varphi(\boldsymbol{A},\ \boldsymbol{BC}) = \boldsymbol{\Omega}, \tag{3.9a}$$
$$\lambda_{mk} b_{mk} = \omega_{kn} c_{kn} = 0. \tag{3.9b}$$

Writing out the gradient $\nabla_{\boldsymbol{B}} D_\varphi(\boldsymbol{A},\ \boldsymbol{BC})$ elementwise, multiplying by $b_{mk}$, and making use of (3.9a,b), we obtain

$$\left[ \big(\zeta(\boldsymbol{BC}) \odot (\boldsymbol{BC} - \boldsymbol{A})\big) \boldsymbol{C}^T \right]_{mk} b_{mk} = \lambda_{mk} b_{mk} = 0,$$

which suggests the iterative scheme

$$b_{mk} \leftarrow b_{mk} \frac{\left[ \big(\zeta(\boldsymbol{BC}) \odot \boldsymbol{A}\big) \boldsymbol{C}^T \right]_{mk}}{\left[ \big(\zeta(\boldsymbol{BC}) \odot \boldsymbol{BC}\big) \boldsymbol{C}^T \right]_{mk}}. \tag{3.10}$$

Proceeding in a similar fashion we obtain a similar iterative formula for $c_{kn}$, which is

$$c_{kn} \leftarrow c_{kn} \frac{\left[ \boldsymbol{B}^T \big(\zeta(\boldsymbol{BC}) \odot \boldsymbol{A}\big) \right]_{kn}}{\left[ \boldsymbol{B}^T \big(\zeta(\boldsymbol{BC}) \odot \boldsymbol{BC}\big) \right]_{kn}}. \tag{3.11}$$

### 3.2.1 Examples of New and Old NNMA Problems as Special Cases

We now illustrate the power of our approach by showing how one can easily obtain iterative update relations for many NNMA problems, including known and new problems. For more examples and further generalizations we refer the reader to [7].

**Lee and Seung's Algorithms.** Let $\alpha \equiv 0$, $\beta \equiv 0$. Now if we set $\varphi(x) = \frac{1}{2} x^2$ or $\varphi(x) = x \log x$, then (3.10) and (3.11) reduce to the Frobenius norm and KL-Divergence update rules originally derived by Lee and Seung [11].

**Elementwise weighted distortion.** Here we wish to minimize $\|\boldsymbol{W} \odot (\boldsymbol{A} - \boldsymbol{BC})\|_F^2$. Using $\boldsymbol{X} \leftarrow \sqrt{\boldsymbol{W}} \odot \boldsymbol{X}$, and $\boldsymbol{A} \leftarrow \sqrt{\boldsymbol{W}} \odot \boldsymbol{A}$ in (3.10) and (3.11) one obtains

$$\boldsymbol{B} \leftarrow \boldsymbol{B} \odot \frac{(\boldsymbol{W} \odot \boldsymbol{A}) \boldsymbol{C}^T}{(\boldsymbol{W} \odot (\boldsymbol{BC})) \boldsymbol{C}^T}, \qquad \boldsymbol{C} \leftarrow \boldsymbol{C} \odot \frac{\boldsymbol{B}^T (\boldsymbol{W} \odot \boldsymbol{A})}{\boldsymbol{B}^T (\boldsymbol{W} \odot (\boldsymbol{BC}))}.$$

These iterative updates are significantly simpler than the PMF algorithms of [13].

**The Multifactor NNMA Problem (new).** The above ideas can be extended to the multifactor NNMA problem that seeks to minimize the following divergence (see [7])

$$D_\varphi(\boldsymbol{A},\ \boldsymbol{B}_1 \boldsymbol{B}_2 \ldots \boldsymbol{B}_R),$$

where all matrices involved are nonnegative. A typical usage of multifactor NNMA problem would be to obtain a three-factor NNMA, namely $\boldsymbol{A} \approx \boldsymbol{RBC}$. Such an approximation is closely tied to the problem of co-clustering [3], and can be used to produce relaxed co-clustering solutions [7].

**Weighted NNMA Problem (new).** We can follow the same derivation method as above (based on KKT conditions) for obtaining multiplicative updates for the weighted NNMA problem:

$$\min D_\varphi(\boldsymbol{A},\ \boldsymbol{W}_1 \boldsymbol{BC} \boldsymbol{W_2}),$$

where $\boldsymbol{W}_1$ and $\boldsymbol{W}_2$ are nonnegative (and nonsingular) weight matrices. The work of [9] is a special case as mentioned in Table 1. Please refer to [7] for more details.

## 4   Experiments and Discussion

We have looked at generic algorithms for minimizing Bregman divergences between the input and its approximation. One important question arises: Which Bregman divergence should one use for a given problem? Consider the following factor analytic model

$$A = BC + N,$$

where $N$ represents some additive noise present in the measurements $A$, and the aim is to recover $B$ and $C$. If we assume that the noise is distributed according to some member of the exponential family, then minimizing the corresponding Bregman divergence [1] is appropriate. For e.g., if the noise is modeled as i.i.d. Gaussian noise, then the Frobenius norm based problem is natural.

Another question is: Which version of the problem we should use, (2.2) or (2.3)? For $\varphi(x) = \frac{1}{2}x^2$, both problems coincide. For other $\varphi$, the choice between (2.2) and (2.3) can be guided by computation issues or sparsity patterns of $A$. Clearly, further work is needed for answering this question in more detail.

Some other open problems involve looking at the class of minimization problems to which the iterative methods of Section 3.2 may be applied. For example, determining the class of functions $h$, for which these methods may be used to minimize $D_\varphi(A, h(BC))$. Other possible methods for solving both (2.2) and (2.3), such as the use of alternating projections (AP) for NNMA, also merit a study.

Our methods for (2.2) decreased the objective function monotonically (by construction). However, we did not demonstrate such a guarantee for the updates (3.10) & (3.11). Figure 1 offers encouraging empirical evidence in favor of a monotonic behavior of these updates. It is still an open problem to formally prove this monotonic decrease. Preliminary results that yield *new* monotonicity proofs for the Frobenius norm and KL-divergence NNMA problems may be found in [7].

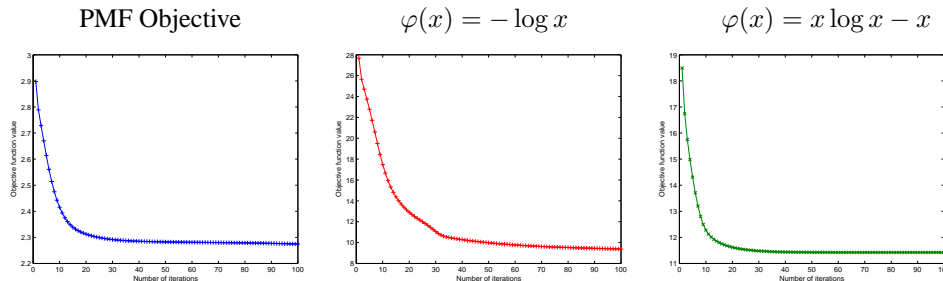

Figure 1: Objective function values over 100 iterations for different NNMA problems. The input matrix $A$ was random $20 \times 8$ nonnegative matrix. Matrices $B$ and $C$ were $20 \times 4$, $4 \times 8$, respectively.

NNMA has been used in a large number of applications, a fact that attests to its importance and appeal. We believe that special cases of our generalized problems will prove to be useful for applications in data mining and machine learning.

## 5   Related Work

Paatero and Tapper [13] introduced NNMA as positive matrix factorization, and they aimed to minimize $\|W \odot (A - BC)\|_F$, where $W$ was a fixed nonnegative matrix of weights. NNMA remained confined to applications in Environmetrics and Chemometrics before pioneering papers of Lee and Seung [11, 12] popularized the problem. Lee and Seung [11] provided simple and efficient algorithms for the NNMA problems that sought to minimize

$\|A - BC\|_{\mathrm{F}}$ and $\mathrm{KL}(A, BC)$. Lee & Seung called these problems *nonnegative matrix factorization* (NNMF), and their algorithms have inspired our generalizations.

NNMA was applied to a host of applications including text analysis, face/image recognition, language modeling, and speech processing amongst others. We refer the reader to [7] for pointers to the literature on various applications of NNMA.

Srebro and Jaakola [15] discuss elementwise weighted low-rank approximations without any nonnegativity constraints. Collins et al. [6] discuss algorithms for obtaining a low rank approximation of the form $A \approx BC$, where the loss functions are Bregman divergences, however, there is no restriction on $B$ and $C$. More recently, Cichocki et al. [4] presented schemes for NNMA with Csiszár's $\varphi$-divergeneces, though rigorous convergence proofs seem to be unavailable. Our approach of Section 3.2 also yields heuristic methods for minimizing Csiszár's divergences.

**Acknowledgments**

This research was supported by NSF grant CCF-0431257, NSF Career Award ACI-0093404, and NSF-ITR award IIS-0325116.

## Footnotes

[1]We use the word *approximation* instead of *factorization* to emphasize the inexactness of the process since, the input $\boldsymbol{A}$ is approximated by $\boldsymbol{BC}$.

# References

[1] A. Banerjee, S. Merugu, I. S. Dhillon, and J. Ghosh. Clustering with Bregman Divergences. In *SIAM International Conf. on Data Mining*, Lake Buena Vista, Florida, April 2004. SIAM.

[2] Y. Censor and S. A. Zenios. *Parallel Optimization: Theory, Algorithms, and Applications*. Numerical Mathematics and Scientific Computation. Oxford University Press, 1997.

[3] H. Cho, I. S. Dhillon, Y. Guan, and S. Sra. Minimum Sum Squared Residue based Co-clustering of Gene Expression data. In *Proc. 4th SIAM International Conference on Data Mining (SDM)*, pages 114–125, Florida, 2004. SIAM.

[4] A. Cichocki, R. Zdunek, and S. Amari. Csiszár's Divergences for Non-Negative Matrix Factorization: Family of New Algorithms. In *6th Int. Conf. ICA & BSS*, USA, March 2006.

[5] M. Collins, R. Schapire, and Y. Singer. Logistic regression, adaBoost, and Bregman distances. In *Thirteenth annual conference on COLT*, 2000.

[6] M. Collins, S. Dasgupta, and R. E. Schapire. A Generalization of Principal Components Analysis to the Exponential Family. In *NIPS 2001*, 2001.

[7] I. S. Dhillon and S. Sra. Generalized nonnegative matrix approximations. Technical report, Computer Sciences, University of Texas at Austin, 2005.

[8] T. Feng, S. Z. Li, H-Y. Shum, and H. Zhang. Local nonnegative matrix factorization as a visual representation. In *Proceedings of the 2nd International Conference on Development and Learning*, pages 178–193, Cambridge, MA, June 2002.

[9] D. Guillamet, M. Bressan, and J. Vitrià. A weighted nonnegative matrix factorization for local representations. In *CVPR*. IEEE, 2001.

[10] P. O. Hoyer. Non-negative sparse coding. In *Proc. IEEE Workshop on Neural Networks for Signal Processing*, pages 557–565, 2002.

[11] D. D. Lee and H. S. Seung. Algorithms for nonnegative matrix factorization. In *NIPS*, pages 556–562, 2000.

[12] D. D. Lee and H. S. Seung. Learning the parts of objects by nonnegative matrix factorization. *Nature*, 401:788–791, October 1999.

[13] P. Paatero and U. Tapper. Positive matrix factorization: A nonnegative factor model with optimal utilization of error estimates of data values. *Environmetrics*, 5(111–126), 1994.

[14] R. T. Rockafellar. *Convex Analysis*. Princeton Univ. Press, 1970.

[15] N. Srebro and T. Jaakola. Weighted low-rank approximations. In *Proc. of 20th ICML*, 2003.

[16] J. A. Tropp. *Topics in Sparse Approximation*. PhD thesis, The Univ. of Texas at Austin, 2004.
